# Constrained Hidden Markov Models

**Sam Roweis**
roweis@gatsby.ucl.ac.uk
Gatsby Unit, University College London

## Abstract

By thinking of each state in a hidden Markov model as corresponding to some spatial region of a fictitious *topology space* it is possible to naturally define neighbouring states as those which are connected in that space. The transition matrix can then be constrained to allow transitions only between neighbours; this means that all valid state sequences correspond to connected paths in the topology space. I show how such *constrained HMMs* can learn to discover underlying structure in complex sequences of high dimensional data, and apply them to the problem of recovering mouth movements from acoustics in continuous speech.

## 1 Latent variable models for structured sequence data

*Structured* time-series are generated by systems whose underlying state variables change in a continuous way but whose state to output mappings are highly nonlinear, many to one and not smooth. Probabilistic unsupervised learning for such sequences requires models with two essential features: latent (hidden) variables and *topology* in those variables. Hidden Markov models (HMMs) can be thought of as dynamic generalizations of discrete state static data models such as Gaussian mixtures, or as discrete state versions of linear dynamical systems (LDSs) (which are themselves dynamic generalizations of continuous latent variable models such as factor analysis). While both HMMs and LDSs provide probabilistic latent variable models for time-series, both have important limitations. Traditional HMMs have a very powerful model of the relationship between the underlying state and the associated observations because each state stores a private distribution over the output variables. This means that any change in the hidden state can cause arbitrarily complex changes in the output distribution. However, it is extremely difficult to capture reasonable dynamics on the discrete latent variable because in principle any state is reachable from any other state at any time step and the next state depends only on the current state. LDSs, on the other hand, have an extremely impoverished representation of the outputs as a function of the latent variables since this transformation is restricted to be global and linear. But it is somewhat easier to capture state dynamics since the state is a multidimensional vector of continuous variables on which a matrix "flow" is acting; this enforces some continuity of the latent variables across time. *Constrained hidden Markov models* address the modeling of state dynamics by building some topology into the hidden state representation. The essential idea is to constrain the transition parameters of a conventional HMM so that the discrete-valued hidden state evolves in a structured way.[1] In particular, below I consider parameter restrictions which constrain the state to evolve as a discretized version of a continuous multivariate variable, i.e. so that it inscribes only connected paths in some space. This lends a physical interpretation to the discrete state trajectories in an HMM.

## 2 An illustrative game

Consider playing the following game: divide a sheet of paper into several contiguous, non-overlapping regions which between them cover it entirely. In each region inscribe a symbol, allowing symbols to be *repeated* in different regions. Place a pencil on the sheet and move it around, reading out (in order) the symbols in the regions through which it passes. Add some *noise* to the observation process so that some fraction of the time incorrect symbols are reported in the list instead of the correct ones. The game is to reconstruct the configuration of regions on the sheet from only such an ordered list(s) of noisy symbols. Of course, the absolute scale, rotation and reflection of the sheet can never be recovered, but learning the essential *topology* may be possible.[2] Figure 1 illustrates this setup.

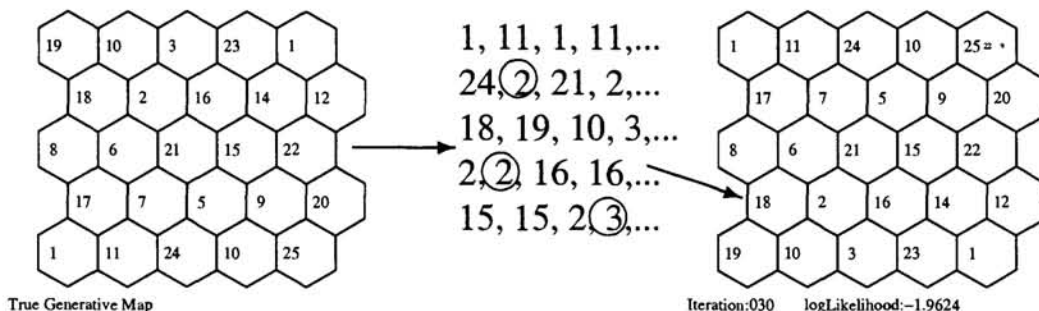

True Generative Map                                    Iteration:030    logLikelihood:−1.9624

**Figure 1:** (**left**) True map which generates symbol sequences by random movement between connected cells. (**centre**) An example noisy output sequence with noisy symbols circled. (**right**) Learned map after training on 3 sequences (with 15% noise probability) each 200 symbols long. Each cell actually contains an entire distribution over all observed symbols, though in this case only the upper right cell has significant probability mass on more than one symbol (see figure 3 for display details).

Without noise or repeated symbols, the game is easy (non-probabilistic methods can solve it) but in their presence it is not. One way of mitigating the noise problem is to do statistical averaging. For example, one could attempt to use the average separation *in time* of each pair of symbols to define a dissimilarity between them. It then would be possible to use methods like multi-dimensional scaling or a sort of *Kohonen mapping though time*[3] to explicitly construct a configuration of points obeying those distance relations. However, such methods still cannot deal with many-to-one state to output mappings (repeated numbers in the sheet) because by their nature they assign a unique spatial location to each symbol.

Playing this game is analogous to doing unsupervised learning on structured sequences. (The game can also be played with continuous outputs, although often high-dimensional data can be effectively clustered around a manageable number of prototypes; thus a vector time-series can be converted into a sequence of symbols.) Constrained HMMs incorporate latent variables with topology yet retain powerful nonlinear output mappings and can deal with the difficulties of noise and many-to-one mappings mentioned above; so they can "win" our game (see figs. 1 & 3). The key insight is that the game generates sequences exactly according to a hidden Markov process whose transition matrix allows only transitions between neighbouring cells and whose output distributions have most of their probability on a single symbol with a small amount on all other symbols to account for noise.

## 3   Model definition: state topologies from cell packings

Defining a constrained HMM involves identifying each state of the underlying (hidden) Markov chain with a spatial cell in a fictitious *topology space*. This requires selecting a *dimensionality d* for the topology space and choosing a *packing* (such as hexagonal or cubic) which fills the space. The number of cells in the packing is equal to the number of states $M$ in the original Markov model. Cells are taken to be all of equal size and (since the scale of the topology space is completely arbitrary) of unit volume. Thus, the packing covers a volume $M$ in topology space with a side length $\ell$ of roughly $\ell = M^{1/d}$. The dimensionality and packing together define a vector-valued function $\mathbf{x}(m)$, $m = 1 \ldots M$ which gives the location of cell $m$ in the packing. (For example, a cubic packing of $d$ dimensional space defines $\mathbf{x}(m+1)$ to be $\left[ m, m/\ell, m/\ell^2, \ldots, m/\ell^{d-1} \right]$ mod $\ell$.) State $m$ in the Markov model is assigned to to cell $m$ in the packing, thus giving it a location $\mathbf{x}(m)$ in the topology space. Finally, we must choose a *neighbourhood rule* in the topology space which defines the neighbours of cell $m$; for example, all "connected" cells, all face neighbours, or all those within a certain radius. (For cubic packings, there are $3^d$-1 connected neighbours and $2d$ face neighbours in a $d$ dimensional topology space.) The neighbourhood rule also defines the boundary conditions of the space – e.g. periodic boundary conditions would make cells on opposite extreme faces of the space neighbours with each other.

The transition matrix of the HMM is now *preprogrammed* to only allow transitions between neighbours. All other transition probabilities are set to zero, making the transition matrix very sparse. (I have set all permitted transitions to be equally likely.) Now, *all valid state sequences in the underlying Markov model represent connected ("city block") paths through the topology space.* Figure 2 illustrates this for a three-dimensional model.

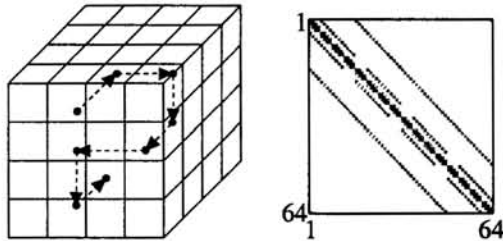

**Figure 2:** **(left)** Physical depiction of the topology space for a constrained HMM with $d$=3,$\ell$=4 and $M$=64 showing an example state trajectory. **(right)** Corresponding transition matrix structure for the 64-state HMM computed using face-centred cubic packing. The gaps in the inner bands are due to edge effects.

## 4   State inference and learning

The constrained HMM has exactly the same inference procedures as a regular HMM: the *forward-backward algorithm* for computing state occupation probabilities and the *Viterbi decoder* for finding the single best state sequence. Once these discrete state inferences have been performed, they can be transformed using the state position function $\mathbf{x}(m)$ to yield probability distributions over the topology space (in the case of forward-backward) or paths through the topology space (in the case of Viterbi decoding). This transformation makes the outputs of state decodings in constrained HMMs comparable to the outputs of inference procedures for continuous state dynamical systems such as Kalman smoothing.

The learning procedure for constrained HMMs is also almost identical to that for HMMs. In particular, the EM algorithm (Baum-Welch) is used to update model parameters. The crucial difference is that the transition probabilities which are precomputed by the topology and packing are never updated during learning. In fact, this makes learning much easier in some cases. Not only do the transition probabilities not have to be learned, but their structure constrains the hidden state sequences in such a way as to make the learning of the output parameters much more efficient when the underlying data really does come from a spatially structured generative model. Figure 3 shows an example of parameter learning for the game discussed above. Notice that in this case, each part of state space had only a single output (except for noise) so the final learned output distributions became essentially minimum entropy. But constrained HMMs can in principle model stochastic or multimodal output processes since each state stores an entire private distribution over outputs.

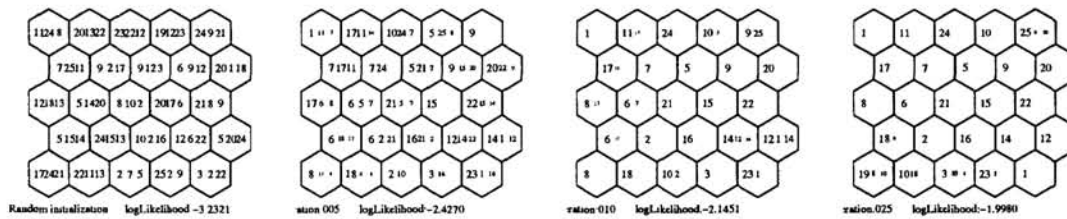

**Figure 3:** Snapshots of model parameters during constrained HMM learning for the game described in section 2. At every iteration each cell in the map has a complete distribution over all of the observed symbols. Only the top three symbols of each cell's histogram are show, with *font size proportional to the square root of probability* (to make ink roughly proportional). The map was trained on 3 noisy sequences each 200 symbols long generated from the map on the left of figure 1 using 15% noise probability. The final map after convergence (30 iterations) is shown on the right of figure 1.

## 5  Recovery of mouth movements from speech audio

I have applied the constrained HMM approach described above to the problem of recovering mouth movements from the acoustic waveform in human speech. Data containing simultaneous audio and articulator movement information was obtained from the University of Wisconsin X-ray microbeam database [9]. Eight separate points (four on the tongue, one on each lip and two on the jaw) located in the midsaggital plane of the speaker's head were tracked while subjects read various words, sentences, paragraphs and lists of numbers. The $x$ and $y$ coordinates (to within about $\pm 1$mm) of each point were sampled at 146Hz by an X-ray system which located gold beads attached to the feature points on the mouth, producing a 16-dimensional vector every 6.9ms. The audio was sampled at 22kHz with roughly 14 bits of amplitude resolution but in the presence of machine noise.

These data are well suited to the constrained HMM architecture. They come from a system whose state variables are known, because of physical constraints, to move in connected paths in a low degree-of-freedom space. In other words the (normally hidden) articulators (movable structures of the mouth), whose positions represent the underlying state of the speech production system,[4] move slowly and smoothly. The observed speech signal—the system's output—can be characterized by a sequence of short-time spectral feature vectors, often known as a *spectrogram*. In the experiments reported here, I have characterized the audio signal using 12 line spectral frequencies (LSFs) measured every 6.9ms (to coincide with the articulatory sampling rate) over a 25ms window. These LSF vectors characterize only the *spectral shape* of the speech waveform over a short time but not its energy. Average energy (also over a 25ms window every 6.9ms) was measured as a separate one dimensional signal. Unlike the movements of the articulators, the audio spectrum/energy can exhibit quite abrupt changes, indicating that the mapping between articulator positions and spectral shape is not smooth. Furthermore, the mapping is many to one: *different* articulator configurations can produce very similar spectra (see below).

The unsupervised learning task, then, is to explain the complicated sequences of observed spectral features (LSFs) and energies as the outputs of a system with a low-dimensional state vector that changes slowly and smoothly. In other words, can we learn the parameters[5] of a constrained HMM such that connected paths through the topology space (state space) generate the acoustic training data with high likelihood? Once this unsupervised learning task has been performed, we can (as I show below) relate the learned trajectories in the topology space to the true (measured) articulator movements.

While many models of the speech production process predict the many-to-one and non-smooth properties of the articulatory to acoustic mapping, it is useful to confirm these features by looking at real data. Figure 4 shows the experimentally observed distribution of articulator configurations used to produce similar sounds. It was computed as follows. All the acoustic and articulatory data for a single speaker are collected together. Starting with some sample called the *key sample*, I find the 1000 samples "nearest" to this key by two measures: articulatory distance, defined using the Mahalanobis norm between two position vectors under the global covariance of all positions for the appropriate speaker, and spectral shape distance, again defined using the Mahalanobis norm but now between two line spectral frequency vectors using the global LSF covariance of the speaker's audio data. In other words, I find the 1000 samples that "look most like" the key sample in mouth shape and that "sound most like" the key sample in spectral shape. I then plot the tongue bead positions of the key sample (as a thick cross), and the 1000 nearest samples by mouth shape (as a thick ellipse) and spectral shape (as dots). The points of primary interest are the dots; they show the distribution of tongue positions used to generate very similar sounds. (The thick ellipses are shown only as a control to ensure that many nearby points to the key sample *do* exist in the dataset.) Spread or multimodality in the dots indicates that many *different* articulatory configurations are used to generate the *same* sound.

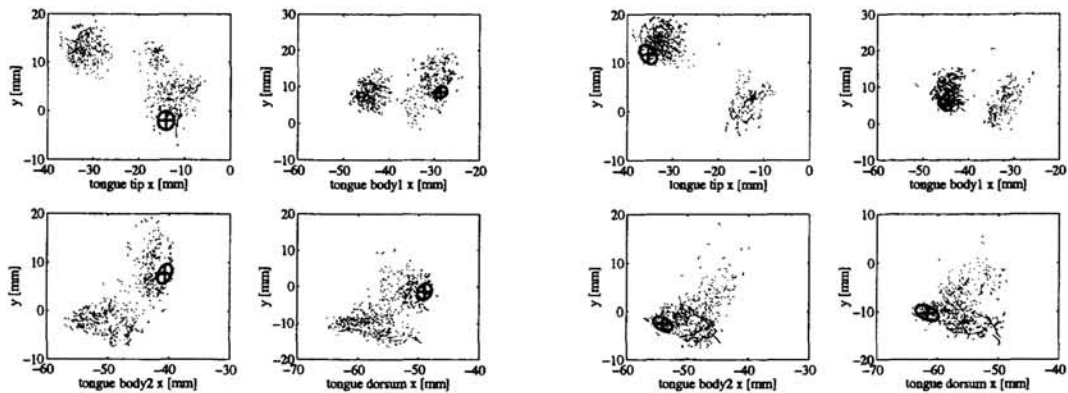

**Figure 4:** Inverse mapping from acoustics to articulation is ill-posed in real speech production data. Each group of four articulator-space plots shows the 1000 samples which are "nearest" to one key sample (thick cross). The dots are the 1000 nearest samples using an acoustic measure based on line spectral frequencies. Spread or multimodality in the dots indicates that many different articulatory configurations are used to generate very similar sounds. Only the positions of the four tongue beads have been plotted. Two examples (with different key samples) are shown, one in the left group of four panels and another in the right group. The thick ellipses (shown as a control) are the two-standard deviation contour of the 1000 nearest samples using an articulatory position distance metric.

Why not do direct supervised learning from short-time spectral features (LSFs) to the articulator positions? The ill-posed nature of the inverse problem as shown in figure 4 makes this impossible. To illustrate this difficulty, I have attempted to recover the articulator positions from the acoustic feature vectors using Kalman smoothing on a LDS. In this case, since we have access to both the hidden states (articulator positions) and the system outputs (LSFs) we can compute the optimal parameters of the model directly. (In particular, the state transition matrix is obtained by regression from articulator positions and velocities at time $t$ onto positions at time $t+1$; the output matrix by regression from articulator positions and velocities onto LSF vectors; and the noise covariances from the residuals of these regressions.) Figure 5b shows the results of such smoothing; the recovery is quite poor.

Constrained HMMs can be applied to this recovery problem, as previously reported [6]. (My earlier results used a small subset of the same database that was not continuous speech and did not provide the hard experimental verification (fig. 4) of the many-to-one problem.)

**Figure 5:** (**A**) Recovered articulator movements using state inference on a constrained HMM. A four-dimensional model with 4096 states was trained on data (all beads) from a single speaker but not including the test utterance shown. Dots show the actual measured articulator movements for a single bead coordinate versus time; the thin lines are estimated movements from the corresponding acoustics. (**B**) Unsuccessful recovery of articulator movements using Kalman smoothing on a global LDS model. All the (speaker-dependent) parameters of the underlying linear dynamical system are known; they have been set to their optimal values using the true movement information from the training data. Furthermore, for this example, the test utterance shown was included in the training data used to estimate model parameters. (**C**) All 16 bead coordinates; all vertical axes are the same scale. Bead names are shown on the left. Horizontal movements are plotted in the left-hand column and vertical movements in the right-hand column. The separation between the two horizontal lines near the centre of the right panel indicates the machine measurement error.

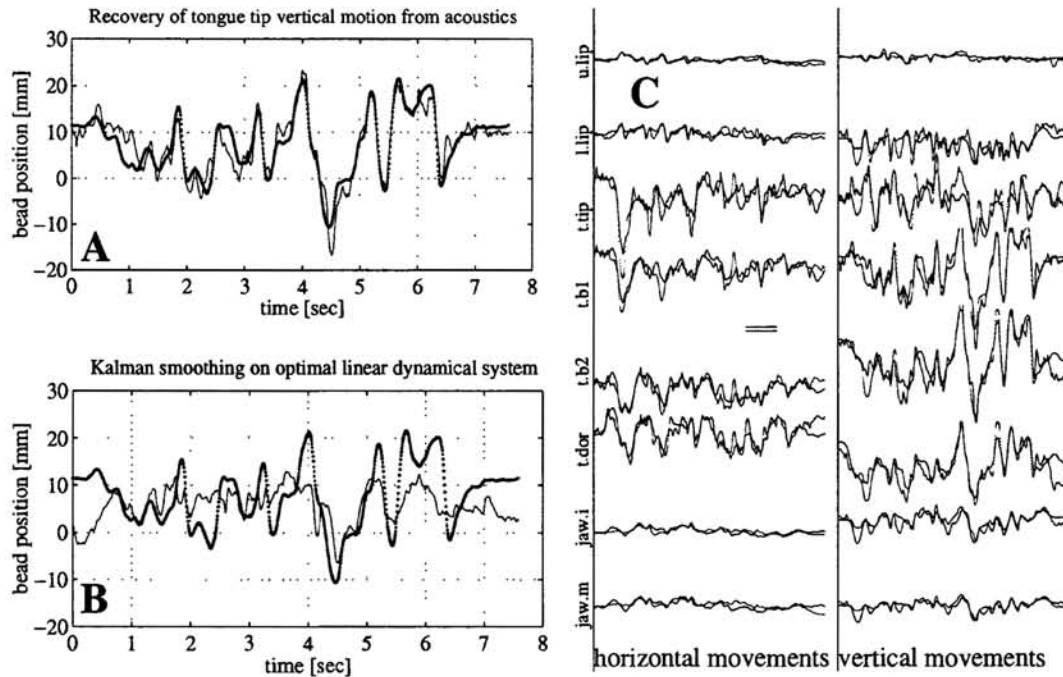

The basic idea is to train (unsupervised) on sequences of acoustic-spectral features and then map the topology space state trajectories onto the measured articulatory movements. Figure 5 shows movement recovery using state inference in a four-dimensional model with 4096 states ($d=4,\ell=8,M=4096$) trained on data (all beads) from a single speaker. (Naive unsupervised learning runs into severe local minima problems. To avoid these, in the simulations shown above, models were trained by slowly annealing two learning parameters[6]: a term $\epsilon^\beta$ was used in place of the zeros in the sparse transition matrix, and $\gamma_t^\beta$ was used in place of $\gamma_t = p(m_t|observations)$ during inference of state occupation probabilities. Inverse temperature $\beta$ was raised from 0 to 1.) To infer a continuous state trajectory from an utterance after learning, I first do Viterbi decoding on the acoustics to generate a discrete state sequence $m_t$ and then interpolate smoothly between the positions $\mathbf{x}(m_t)$ of each state.

After unsupervised learning, a single linear fit is performed between these continuous state trajectories and actual articulator movements on the training data. (The model cannot discover the units system or axes used to represent the articulatory data.) To recover articulator movements from a previously unseen test utterance, I infer a continuous state trajectory as above and then apply the single linear mapping (learned only once from the training data).

## 6   Conclusions, extensions and other work

By enforcing a simple constraint on the transition parameters of a standard HMM, a link can be forged between discrete state dynamics and the motion of a real-valued state vector in a continuous space. For complex time-series generated by systems whose underlying latent variables do in fact change slowly and smoothly, such constrained HMMs provide a powerful unsupervised learning paradigm. They can model state to output mappings that are highly nonlinear, many to one and not smooth. Furthermore, they rely only on well understood learning and inference procedures that come with convergence guarantees.

Results on synthetic and real data show that these models can successfully capture the low-dimensional structure present in complex vector time-series. In particular, I have shown that a speaker dependent constrained HMM can accurately recover articulator movements from continuous speech to within the measurement error of the data. This acoustic to articulatory inversion problem has a long history in speech processing (see e.g. [7] and references therein). Many previous approaches have attempted to exploit the smoothness of articulatory movements for inversion or modeling: Hogden *et.al* (e.g. [4]) provided early inspiration for my ideas, but do not address the many-to-one problem; Simon Blackburn [1] has investigated a *forward* mapping from articulation to acoustics but does not explicitly attempt inversion; early work at Waterloo [5] suggested similar constraints for improving speech recognition systems but did look at real articulatory data, more recent work at Rutgers [2] developed a very similar system much further with good success. Perpiñán [3], considers a related problem in sequence learning using EPG speech data as an example.

While in this note I have described only "diffusion" type dynamics (transitions to all neighbours are equally likely) it is also possible to consider *directed flows* which give certain neighbours of a state lower (or zero) probability. The left-to-right HMMs mentioned earlier are an example of this for one-dimensional topologies. For higher dimensions, flows can be derived from discretization of matrix (linear) dynamics or from other physical/structural constraints. It is also possible to have many connected local flow regimes (either diffusive or directed) rather than one global regime as discussed above; this gives rise to *mixtures* of constrained HMMs which have block-structured rather than banded transition matrices. Smyth [8] has considered such models in the case of one-dimensional topologies and directed flows; I have applied these to learning character sequences from English text. Another application I have investigated is map learning from multiple sensor readings. An explorer (robot) navigates in an unknown environment and records at each time many local measurements such as altitude, pressure, temperature, humidity, etc. We wish to reconstruct from only these sequences of readings the topographic maps (in each sensor variable) of the area as well as the trajectory of the explorer. A final application is tracking (inferring movements) of articulated bodies using video measurements of feature positions.

**References**
[1] S. Blackburn & S. Young. *ICSLP 1996*, Philadephia, v.2 pp.969–972
[2] S. Chennoukh *et.al*, *Eurospeech 1997*, Rhodes, Greece, v.1 pp.429–432
[3] M. Carreira-Perpiñán. *NIPS'12*, 2000. (This volume.)
[4] D. Nix & J. Hogden. *NIPS'11*, 1999, pp.744–750
[5] G. Ramsay & L. Deng. *J. Acoustical Society of America*, 95(5), 1994, p.2873
[6] S. Roweis & A. Alwan. *Eurospeech 1997*, Rhodes, Greece, v.3 pp.1227–1230
[7] J. Schroeter & M. Sondhi. *IEEE Trans.Speech & Audio Processing*, 2(1p2), 1994, pp.133–150
[8] P. Smyth. *NIPS'9*, 1997, pp.648–654
[9] J. Westbury. X-ray microbeam speech production database user's handbook version 1.0.
   University of Wisconsin, Madison, June 1994.

## Footnotes

[1] A standard trick in traditional speech applications of HMMs is to use "left-to-right" transition matrices which are a special case of the type of constraints investigated in this paper. However, left-to-right (Bakis) HMMs force state trajectories that are inherently one-dimensional and uni-directional whereas here I also consider higher dimensional topology and free omni-directional motion.

[2] The observed symbol sequence must be "informative enough" to reveal the map structure (this can be quantified using the idea of *persistent excitation* from control theory).

[3] Consider a network of units which compete to explain input data points. Each unit has a position in the output space as well as a position in a lower dimensional topology space. The winning unit has its position in output space updated towards the data point; but also the recent (in time) winners have their positions in topology space updated towards the topology space location of the current winner. Such a rule works well, and yields topological maps in which *nearby units code for data that typically occur close together in time*. However it cannot learn many-to-one maps in which more than one unit at different topology locations have the same (or very similar) outputs.

[4]Articulator positions do not provide complete state information. For example, the excitation signal (voiced or unvoiced) is not captured by the bead locations. They do, however, provide much important information; other state information is easily accessible directly from acoustics.

[5]Model structure (dimensionality and number of states) is currently set using cross validation.

[6]An easier way (which I have used previously) to find good minima is to initialize the models using the articulatory data themselves. This does not provide as impressive "structure discovery" as annealing but still yields a system capable of inverting acoustics into articulatory movements on previously unseen test data. First, a constrained HMM is trained on just the articulatory movements; this works easily because of the natural geometric (physical) constraints. Next, I take the distribution of acoustic features (LSFs) over all times (in the training data) when Viterbi decoding places the model in a particular state and use those LSF distributions to initialize an equivalent acoustic constrained HMM. This new model is then retrained until convergence using Baum-Welch.
